# Bias, Variance and the Combination of Least Squares Estimators

**Ronny Meir**
Faculty of Electrical Engineering
Technion, Haifa 32000
Israel
`rmeir@ee.technion.ac.il`

## Abstract

We consider the effect of combining several least squares estimators on the expected performance of a regression problem. Computing the exact bias and variance curves as a function of the sample size we are able to quantitatively compare the effect of the combination on the bias and variance separately, and thus on the expected error which is the sum of the two. Our exact calculations, demonstrate that the combination of estimators is particularly useful in the case where the data set is small and noisy and the function to be learned is unrealizable. For large data sets the single estimator produces superior results. Finally, we show that by splitting the data set into several independent parts and training each estimator on a different subset, the performance can in some cases be significantly improved.

**Key words:** Bias, Variance, Least Squares, Combination.

## 1 INTRODUCTION

Many of the problems related to supervised learning can be boiled down to the question of balancing bias and variance. While reducing bias can usually be accomplished quite easily by simply increasing the complexity of the class of models studied, this usually comes at the expense of increasing the variance in such a way that the overall expected error (which is the sum of the two) is often increased.

Thus, many efforts have been devoted to the issue of decreasing variance, while attempting to keep the concomitant increase in bias as small as possible. One of the methods which has become popular recently in the neural network community is variance reduction by combining estimators, although the idea has been around in the statistics and econometrics literature at least since the late sixties (see Granger 1989 for a review). Nevertheless, it seems that not much analytic work has been devoted to a detailed study of the effect of noise and an effectively *finite* sample size on the bias/variance balance. It is the explicit goal of this paper to study in detail a simple problem of linear regression, where the full bias/variance curve can be computed exactly for *any* effectively finite sample size and noise level. We believe that this simple and exactly solvable model can afford us insight into more complex non-linear problems, which are at the heart of much of the recent work in neural networks.

A further aspect of our work is related to the question of the partitioning of the data set between the various estimators. Thus, while most studies assume the each estimator is trained on the complete data set, it is possible to envisage a situation where the data set is broken up into several subsets, using each subset of data to form a different estimator. While such a scheme seems wasteful from the bias point of view, we will see that in fact it produces superior forecasts in some situations. This, perhaps suprising, result is due to a large decrease in variance resulting from the independence of the estimators, in the case where the data subsets are independent.

## 2   ON THE COMBINATION OF ESTIMATORS

The basic objective of regression is the following: given a finite training set, $D$, composed of $n$ input/output pairs, $D = \{(\mathbf{x}_\mu, y_\mu)\}_{\mu=1}^n$, drawn according to an *unkown* distribution $P(\mathbf{x}, y)$, find a function ('estimator'), $f(\mathbf{x}; D)$, which 'best' approximates $y$. Using the popular mean-squared error criterion and taking expectations with respect to the data distribution one finds the well-known separation of the error into a bias and variance terms respectively (Geman et al. 1992)

$$\mathcal{E}(\mathbf{x}) = (E_D f(\mathbf{x}; D) - E[y|\mathbf{x}])^2 + E_D \left[ f(\mathbf{x}; D) - E_D f(\mathbf{x}; D) \right]^2 . \tag{1}$$

We consider a data source of the form $y = g(\mathbf{x}) + \eta$, where the 'target' function $g(\mathbf{x})$ is an unknown (and potentially non-linear) function and $\eta$ is a Gaussian random variable with zero mean and variance $\sigma^2$. Clearly then $E[y|\mathbf{x}] = g(\mathbf{x})$.

In the usual scenario for parameter estimation one uses the complete data set, $D$, to form an estimator $f(\mathbf{x}; D)$. In this paper we consider the case where the data set $D$ is broken up into $K$ subsets (not necessarily disjoint), such that $D = \cap_{k=1}^K D^{(k)}$, and a separate estimator is found for each subset. The full estimator is then given by the linear combination (Granger 1989)

$$f(\mathbf{x}; D) = \sum_{k=1}^K b_k f_k(\mathbf{x}; D^{(k)}) . \tag{2}$$

The optimal values of the parameters $b_k$ can be easily obtained if the data distribution, $P(\mathbf{x}, y)$, is known, by simply minimizing the mean-squared error (Granger

1989). In the more typical case where this distribution is unkown, one may resort to other schemes such as least-squares fitting for the parameter vector $\mathbf{b} = \{b_1, \ldots, b_K\}$. The bias and variance of the combined estimator can be simply expressed in this case, and are given by

$$B(\mathbf{x}; g) = \left( \sum_{k=1}^{K} b_k \overline{f_k}(\mathbf{x}) - g(\mathbf{x}) \right)^2 \quad ; \quad V(\mathbf{x}; g) = \sum_{k,k'} b_k b_{k'} \left\{ \overline{f_k f_{k'}}(\mathbf{x}) - \overline{f_k}(\mathbf{x}) \overline{f_{k'}}(\mathbf{x}) \right\}$$

(3)

where the overbars denote an average with respect to the data. It is immediately apparent that the variance term is composed of two contributions. The first term, corresponding to $k = k'$, simply computes a weighted average of the single estimator variances, while the second term measures the average covariance between the different estimators. While the first term in the variance can be seen to decay as $1/K$ in the case where all the weights $b_k$ are of the same order of magnitude, the second term is finite *unless* the covariances between estimators are very small. It would thus seem beneficial to attempt to make the estimators as weakly correlated as possible in order to decrease the variance. Observe that in the extreme case where the data sets are independent of each other, the second term in the variance vanishes identically. Note that the bias term depends only on single estimator properties and can thus be computed from the theory of the single estimator. As mentioned above, however, the second term in the variance expression explicitly depends on correlations between the different estimators, and thus requires the computation of quantities beyond those of single estimators.

## 3   THE SINGLE LINEAR ESTIMATOR

Before considering the case of a combination of estimators, we first review the case of a single *linear* estimator, given by $f(\mathbf{x}; D) = \hat{\mathbf{w}}^T \cdot \mathbf{x}$, where $\hat{\mathbf{w}}$ is estimated from the data set $D$. Following Bös et al. (1993) we further assume that the data arises through an equation of the form $y = g(\mathbf{x}) + \eta$ with $g = g(\mathbf{w}_0^T \cdot \mathbf{x})$. Looking back at equations (3) it is clear that the bias and variance are explicit functions of $\mathbf{x}$ and the weight vector $\mathbf{w}_0$. In order to remove the explicit dependence we compute in what follows expectations with respect to the probability distribution of $\mathbf{x}$ and $\mathbf{w}_0$, denoted respectively by $E_p[\cdot]$ and $E_0[\cdot]$. Thus, we define the averaged bias and variance by $B = E_0 E_p[B(\mathbf{x}; \mathbf{w}_0)]$ and $V = E_0 E_p[V(\mathbf{x}; \mathbf{w}_0)]$ and the expected error is then $\mathcal{E} = B + V$.

In this work we consider *least-squares* estimation which corresponds to minimizing the empirical error, $\mathcal{E}_{\text{emp}}(\mathbf{w}, D) = \|X\mathbf{w} - Y\|^2$, where $X$ is the $n \times d$ data matrix, $Y$ is the $n \times 1$ output vector and $\mathbf{w}$ is a $d \times 1$ weight vector. The components of the 'target' vector $Y$ are given by $y_\mu = g(\mathbf{w}_0^T \cdot \mathbf{x}_\mu) + \eta_\mu$ where $\eta_\mu$ are i.i.d normal random variables with zero mean and variance $\sigma^2$. Note that while we take the estimator itself to be linear we allow the target function $g(\cdot)$ to be non-linear. This is meant to model the common situation where the model we are trying to fit is inadequate, since the correct model (even it exists) is usually unkown.

Thus, the least squares estimator is given by $\hat{\mathbf{w}} \in \arg\min_{\mathbf{w}} \mathcal{E}_{\text{emp}}(\mathbf{w}, D)$. Since in this case the error-function is quadratic it possesses either a unique global minimum

or a degenerate manifold of minima, in the case where the Hessian matrix, $X^T X$, is singular.

The solution to the least squares problem is well known (see for example Scharf 1991), and will be briefly summarized. When the number of examples, $n$, is smaller than the input dimension, $d$, the problem is underdetermined and there are many solutions with *zero* empirical error. The solutions can be written out explicitly in the form

$$\hat{w} = X^T(XX^T)^{-1}Y + (I - X^T(XX^T)^{-1}X) V \qquad (n < d), \qquad (4)$$

where $V$ is an *arbitrary* $d$-dimensional vector. It should be noted that any vector **w** satisfying this equation (and thus any least-squares estimator) becomes singular as $n$ approaches $d$ from below, since the matrix $XX^T$ becomes singular. The minimal norm solution, often referred to as the *Moore-Penrose* solution, is given in this case by the choice $V = 0$. It is common in the literature to neglect the study of the underdetermined regime since the solution is not unique in this case. We however will pay specific attention to this case, corresponding to the often prevalent situation where the amount of data is small, attempting to show that the combination of estimators approach can significantly improve the quality of predictors in this regime. Moreover, many important inverse problems in signal processing fall into this category (Scharf 1991).

In the overdetermined case, $n > d$ (assuming the matrix $X$ to be of full rank), a zero error solution is possible only if the function $g(\cdot)$ is linear and there is no noise, namely $E[\eta^2] = 0$. In any other case, the problem becomes *unrealizable* and the minimum error is non-zero. In any event, in this regime the *unique* solution minimizing the empirical error is given by

$$\hat{\mathbf{w}} = (X^T X)^{-1} X^T Y \qquad (n > d). \qquad (5)$$

It is eay to see that this estimator is unbiased for linear $g(\cdot)$.

In order to compute the bias and variance for this model we use Eqs. (3) with $K = 1$ and $b_k = 1$. In order to actually compute the expectations with respect to **x** and the weight vector $\mathbf{w}_0$ we assume in what follows that the random vector **x** is distributed according to a multi-dimensional normal distributions of zero mean and covariance matrix $(1/d)I$. The vector $\mathbf{w}_0$ is similarly distributed with unit covariance matrix. The reason for the particular scaling chosen for the covariance matrices will become clear below. In the remainder of the paper we will be concerned with exact calculations in the so called *thermodynamic limit*: $n, d \to \infty$ and $\alpha = n/d$ finite. This limit is particularly useful in that the central limit theorem allows one to make precise statements about the behavior of the system, for an *effectively finite* sample size, $\alpha$. We note in passing that in the thermodynamic limit, $d \to \infty$, we have $\sum_i x_i^2 \to 1$ with probability 1 and similarly for $(1/d)\sum_i w_{0i}^2$. Using these simple distributions we can, after some algerbra, directly compute the bias and variance. Denoting $R = E_0[\overline{\mathbf{w}}^T \cdot \mathbf{w}_0]$, $r = E_0\|\overline{\mathbf{w}}\|^2$, $Q = E_0\|\hat{\mathbf{w}}\|^2$, one can show that the bias and variance are given by

$$B = r - 2\overline{ug}R + \overline{g^2} \quad ; \quad V = Q - r. \qquad (6)$$

In the above equations we have used $\overline{g^2} = \int Du g^2(u)$ and $\overline{ug} = \int Du \, ug(u)$ where the Gaussian measure $Du$ is defined by $Du = (e^{-u^2/2}/\sqrt{2\pi})du$. We note in passing

that the same result is obtained for any i.i.d variables, $x_i$, with zero mean and variance $1/d$. We thus note that a complete calculation of the expected bias and variance requires the explicit computation of the variables $R$, $r$ and $Q$ defined above. In principle, with the explicit expressions (4) and (5) at hand one may proceed to compute all the quantities relevant to the evaluation of the bias and variance. Unfortunately, it turns out that a direct computation of $r$, $R$ and $Q$ using these expressions is a rather difficult task in the theory of random matrices, keeping in mind the potential non-linearity of the function $g(\cdot)$. A way to solve the problem can be undertaken via a slightly indirect route, using tools from statistical physics. The variables $R$ and $Q$ above have been recently computed by Bös et al. (1993) using replicas and by Opper and Kinzel (1994) by a direct calculation. The variable $r$ can be computed along similar lines resulting in the following expressions for the bias and variance (given for brevity for the Moore-Penrose solution):

$$\alpha < 1: \quad B = \overline{g^2} - \alpha(2-\alpha)\overline{ug}^2 \ , \quad V = \frac{\alpha}{1-\alpha}\left[\overline{g^2} + \sigma^2 - \alpha(2-\alpha)\overline{ug}^2\right]$$

$$\alpha > 1: \quad B = \overline{g^2} - \overline{ug}^2 \quad\quad , \quad V = \frac{1}{\alpha-1}\left[\overline{g^2} + \sigma^2 - \overline{ug}^2\right] \quad (7)$$

We see from this solution that for $\alpha > 1$ the bias is constant, while the variance is monotonically decreasing with the sample size $\alpha$. For $\alpha < 1$, there are of course multiple solutions corresponding to different normalizations $Q$. It is easy to see, however, that the Moore-Penrose solution, gives rise to the smallest variance of all least-squares estimators (the bias is unaffected by the normalization of the solution). The expected (or generalization) error is given simply by $\mathcal{E} = B + V$, and is thus smallest for the Moore-Penrose solution. Note that this latter result is independent of whether the function $g(\cdot)$ is linear or not. We note in passing that in the simple case where the target function $g(\cdot)$ is *linear* and the data is noise-free ($\sigma^2 = 0$) one obtains the particularly simple result $\mathcal{E} = 1 - \alpha$ for $\alpha < 1$ and $\mathcal{E} = 0$ above $\alpha = 1$. Note however that in any other case the expected error is a non-linear function of the normalized sample size $\alpha$.

## 4   COMBINING LINEAR ESTIMATORS

Having summarized the situation in the case of a single estimator, we proceed to the case of $K$ linear estimators. In this case we assume that the complete data set, $D$, is broken up into $K$ subsets $D^{(k)}$ of size $n_k = \alpha_k d$ each. In particular we consider two extreme situations: (i) The data sets are independent of each other, namely $D^{(k)} \cap D^{(k')} = \emptyset$ for all $k$, and (ii) $D^{(k)} = D$ for all $k$. We refer to the first situation as the non-overlapping case, while to the second case where all estimators are trained on the same data set as the fully-overlapping case. Denoting by $\hat{\mathbf{w}}^{(k)}$ the $k$'th least-squares estimator, based on training set $D^{(k)}$, we define the following quantities:

$$R^{(k)} = E_0\left[\overline{\hat{\mathbf{w}}^{(k)T} \cdot \mathbf{w}_0}\right] \ , \ r^{(k,k')} = E_0\left[\overline{\hat{\mathbf{w}}^{(k)T} \cdot \hat{\mathbf{w}}^{(k')}}\right] \ , \ \rho^{(k,k')} = E_0\left[\overline{\hat{\mathbf{w}}^{(k)T} \cdot \hat{\mathbf{w}}^{(k')}}\right] .$$

$$(8)$$

Making use of Eqs. (3) and the probability distribution of $\mathbf{w}_0$ one then straighfor-

wardly finds for the mixture estimator

$$B_K = \sum_{k,k'} b_k b_{k'} r^{(k,k')} - 2\overline{ug} \sum_k b_k R^{(k)} + \overline{g^2},$$

$$V_K = \sum_k b_k^2 \left[ \rho^{(k,k)} - r^{(k,k)} \right] + \sum_{k \neq k'} b_k b_{k'} \left[ \rho^{(k,k')} - r^{(k,k')} \right]. \tag{9}$$

Now, the computation of the parameters $\rho^{(k,k)}$ and $R^{(k)}$ is identical to the case of a single estimator and can be directly read from the results of Bös et al. (1993) with the single modification of rescaling the sample size $\alpha$ to $\alpha_k$. The only interaction between estimators arises from the terms $\rho^{(k,k')}$ and $r^{(k,k')}$ which couple two different estimators. Note however that $r^{(k,k')}$ is independent of the degree of overlap between the data sets $D^{(k)}$ since each estimator is averaged independently. In fact, a straighforward, if lengthy, calculation leads to the conclusion that in general $r^{(k,k')} = R^{(k)} R^{(k')}$. For the case where the data sets are independent of each other, namely $D^{(k)} \cap D^{(k')} = \emptyset$ it is obvious that $\rho^{(k,k')} = r^{(k,k')}$. One then finds $\rho^{(k,k')} = r^{(k,k')} = R^{(k)} R^{(k')}$ for independent data sets. In the case where the data sets are identical, $D^{(k)} = D$ for all $k$, the variable $\rho^{(k,k')}$ has been computed by Bös et al. (1993). At this point we proceed by discussing individually the two data generation scenarios described in section 1.

## 4.1 NON-OVERLAPPING DATA SETS

As shown above, since the data sets are independent of each other one has in this case $\rho^{(k,k')} = r^{(k,k')}$, implying that the second term in the expression for the variance in Eq. (9) vanishes. Specializing now to the case where the sizes of the data sets are equal, namely $\alpha_k = n_k/d = \alpha/K$ for all $k$, we expect the variables $R^{(k)}$ and $Q^{(k)}$ to be independent of the specific index $k$. In the particular case of a *uniform mixture* ($b_k = 1/K$ for all $k$) we obtain

$$B_K^{(u)} = R_K^2 - 2\overline{ug} R_K + \overline{g^2} \quad ; \quad V_K^{(u)} = (Q_K - R_K^2)/K, \tag{10}$$

where the subscript $K$ indicates that the values of the corresponding variables must be taken with respect to $\alpha_K = \alpha/K$. The specific values of the parameters $R_K$ and $Q_K$ can be found directly from the solution of Bös et al. (1993) by replacing $\alpha$ by $\alpha/K$ and dividing the variance term by a further $K$ factor. Since the bias is a non-increasing function $\alpha$ it is always increased (or at best unchanged) by combining estimators from non-overlapping data subsets. It can also be seen from the above equation that as $K$ increases the variance can be made arbitrarily small, albeit at the expense of increasing the bias. It can thus be expected that for any sample size $\alpha$ there is an optimal number of estimators minimizing the expected error, $\mathcal{E} = B + V$. In fact, for small $\alpha$ one may expand the equations for the bias and variance obtaining the expected error from which it is easy to see that the optimal value of $K$ is given by $K^* = (\overline{g^2} + \sigma^2)/\overline{ug}^2$. As could be expected we see that the optimal value of $K$ scales with the noise level $\sigma^2$, since the effect of combining multiple independent estimators is to reduce the effective noise variance by a factor of $K$. As the sample size $\alpha$ increases and outweighs the effect of the noise, we find that the bias increase due to the data partitioning dominates the

decrease in variance and the combined estimator performs more and more poorly relative to the single estimator. Finally, for $\alpha > K$ we find the choice $K = 1$ always yields a lower expected error. Thus, while we have shown that for small sample sizes the effect of splitting the data set into independent subsets is helpful, this is no longer the case if the sample size is sufficiently large, in which case a single estimator based on the complete data set is superior. For $\alpha \to \infty$, however, one finds that all uniform mixtures converge (to leading order) at the same rate, namely $\mathcal{E}_K(\alpha) \approx \mathcal{E}_\infty + (\mathcal{E}_\infty + \sigma^2)/\alpha$ , where $\mathcal{E}_\infty = \overline{g^2} - \overline{ug}^2$. For finite values of $\alpha$, however, the value of $K$ has a strong effect on the quality of the stimator.

## 4.2 FULLY OVERLAPPING DATA SETS

We focus now on the case where all estimators are formed from the same data set $D$, namely $D^{(k)} = D$ for all $k$. Since there is a unique solution to the least-squares estimation problem for $\alpha > 1$, all least-squares estimators coincide in this regime. Thus, we focus here on the case $\alpha < 1$, where multiple least-squares estimators coexist. We further assume that only mixtures of estimators of the same norm $Q$ are allowed. We obtain for the uniform mixture

$$B_K^{(u)} = R^2 - 2\overline{ug}R + \overline{g^2} \quad ; \quad V_K^{(u)} = (Q - R^2) - (Q - q)(1 - \frac{1}{K}) \quad (\alpha < 1) \quad (11)$$

Clearly the expression for the bias in this case is identical to that obtained for the single estimator, since all estimators are based on the same data set and the bias term depends only on single estimator properties. The variance term, however, is modified due to the correlation between the estimators expressed through the variable $\rho^{(k,k')}$. Since the variance for the case of a single estimator is $Q - R^2$ and since $q \leq Q$ it is clear in this case that the variance is reduced while the bias remains unchanged. Thus we conclude that the mixture of estimators in this case indeed produces superior performance to that of the single estimator. However, it can be seen that in the case of the Moore-Penrose solution, corresponding to choosing the smallest possible norm $Q$, the expected error is minimal. We thus conclude that for $\alpha < 1$ the Moore-Penrose pseudo-inverse solution yields the lowest expected error, and this *cannot* be improved on by combining least-squares estimators obtained from the full data set $D$.

Recall that we have shown in the previous section that (for small and noisy data sets) combining estimators formed using non-overlapping data subsets produced results superior to those of any single estimator trained on the complete data set. An interesting conclusion of these results is that splitting the data set into non-overlapping subsets is a better strategy than training each estimator with the full data. As mentioned previously, the basic reason for this is the independence of the estimators formed in this fashion, which helps to reduce the variance term more drastically than in the case where the estimators are dependent (having been exposed to overlapping data sets).

## 5 CONCLUSIONS

In this paper we have studied the effect of combining different estimators on the performance of linear regression. In particular we have focused on the case of linear

least-squares estimation, computing exactly the full bias and variance curves for the case where the input dimension is very large (the so called thermodynamic limit). While we have focused specifically on the case of linear estimators, it should not be hard to extend these results to simple non-linear functions of the form $f(\mathbf{w}^T \cdot \mathbf{x})$ (see section 2). The case of a combination of more complex estimators (such as multi-layered neural networks) is much more demanding, as even the case of a single such network is rather difficult to analyzes.

Several positive conclusions we can draw from our study are the following. First, the general claim that combining experts is always helpful is clearly fallacious. While we have shown that combining estimators is beneficial in some cases (such as small noisy data sets), this is not the case in general. Second, we have shown that in some situations (specifically unrealizable rules and small sample size) it is advantageous to split the data into several non-overlapping subsets. It turns out that in this case the decrease in variance resulting from the independence of the different estimators, is larger than the concomitant increase in bias. It would be interesting to try to generalize our results to the case where the data is split in a more efficient manner. Third, our results agree with the general notion that when attempting to learn an unrealizable function (whether due to noise or to a mismatch with the target function) the best option is to learn with errors.

Ultimately one would like to have a general theory for combining empirical estimators. Our work has shown that the effect of noise and finite sample size is expected to produce non-trivial effects which are impossible to observe when considering only the asymptotic limit.

## Acknowledgements

The author thanks Manfred Opper for a very helpful conversation and the Ollendorff center of the Electrical Engineering department at the Technion for financial support.

## References

S. Bös., W. Kinzel and M. Opper 1993, The generalization ability of perceptrons with continuous outputs, *Phys. Rev. A* 47:1384-1391.

S. Geman, E. Bienenstock and R. Dorsat 1992, Neural networks and the bias/variance dilemma, *Neural Computation* 4:1-58.

C.W.J. Granger 1989, Combining forecasts - twenty years later, *J. of Forecast.* 8:167-173.

M. Opper and W. Kinzel 1994, Statistical mechanics of generalization, in *Physics of Neural networks*, van Hemmen, J.S., E. Domany and K. Schulten eds., Springer-Verlag, Berlin.

L.L. Scharf, 1991 *Statistical Signal Processing: Detection, Estimation and Time Series Analysis*, Addison-Wesley, Massachusetts.